# Noise and the two-thirds power law

**Uri Maoz**[1,2,3]**, Elon Portugaly**[3]**, Tamar Flash**[2] **and Yair Weiss**[3,1]

[1] Interdisciplinary Center for Neural Computation, The Hebrew University of Jerusalem, Edmond Safra Campus, Givat Ram Jerusalem 91904, Israel; [2] Department of Computer Science and Applied Mathematics, The Weizmann Institute of Science, PO Box 26 Rehovot 76100, Israel; [3] School of Computer Science and Engineering, The Hebrew University of Jerusalem, Edmond Safra Campus, Givat Ram Jerusalem 91904, Israel

## Abstract

The two-thirds power law, an empirical law stating an inverse non-linear relationship between the tangential hand speed and the curvature of its trajectory during curved motion, is widely acknowledged to be an invariant of upper-limb movement. It has also been shown to exist in eye-motion, locomotion and was even demonstrated in motion perception and prediction. This ubiquity has fostered various attempts to uncover the origins of this empirical relationship. In these it was generally attributed either to smoothness in hand- or joint-space or to the result of mechanisms that damp noise inherent in the motor system to produce the smooth trajectories evident in healthy human motion.

We show here that white Gaussian noise also obeys this power-law. Analysis of signal and noise combinations shows that trajectories that were synthetically created not to comply with the power-law are transformed to power-law compliant ones after combination with low levels of noise. Furthermore, there exist colored noise types that drive non-power-law trajectories to power-law compliance and are not affected by smoothing. These results suggest caution when running experiments aimed at verifying the power-law or assuming its underlying existence without proper analysis of the noise. Our results could also suggest that the power-law might be derived not from smoothness or smoothness-inducing mechanisms operating on the noise inherent in our motor system but rather from the correlated noise which is inherent in this motor system.

## 1  Introduction

A number of regularities have been empirically observed for the motion of the end-point of the human upper-limb during curved and drawing movements. One of these has been termed "the two-thirds power law" ([1]). It can be formulated as:

$$v(t) = \text{const} \cdot \kappa(t)^{\beta} \tag{1}$$

or, in log-space

$$\log\left(v(t)\right) = \text{const} + \beta \log\left(\kappa(t)\right) \tag{2}$$

where $v$ is the tangential end-point speed, $\kappa$ is the instantaneous curvature of the path, and $\beta$ is approximately $-\frac{1}{3}$. The various studies that lend support to this power-law go beyond its simple verification. There are those that suggest it as a tool to extract natural segmentation into primitives of complex movements ([2], [3]). Others show the development of the power-law with age for children ([4]). There is also research that suggests it appears for three-dimensional (3D) drawings under isometric force conditions ([5]). It was even found in neural population coding in the monkey motor brain area controlling the hand ([6]). Other studies have located the power-law elsewhere than the hand. It was found to apply in eye-motion ([7]) and even in motion perception ([8],[9]) and movement prediction based on biological motion ([10]). Recent studies have also found it in locomotion ([11]). This power-law has thus been widely accepted as an important invariant in biological movement trajectories, so much so that it has become an evaluation criterion for the quality of models (e.g. [12]).

This has motivated various attempts to find some deeper explanation that supposedly underlies this regularity. The power-law was shown to possibly be a result of minimization of jerk ([13],[14]), jerk along a predefined path ([15]), or endpoint variability due to noise inherent in the motor system ([12]). Others have claimed that it stems from forward kinematics of sinusoidal movements at the joints ([16]). Another explanation has to do with the mathematically interesting fact that motion according to the power-law maintains constant affine velocity ([17],[18]).

We were thus very much surprised by the following:

**Observation**: Given a time series $(x_i, y_i)_{i=1}^{n}$ in which $x_i, y_i \sim N(0,1)$ i.i.d.$(x_i, x_j, y_i, y_j)$ for $i \neq j$, and assuming that the series is of equal time intervals, calculate $\kappa$ and $v$ in order to obtain $\beta$ from the linear regression of $\log(v)$ versus $\log(\kappa)$. The linear regression plot of $\log(v)$ versus $\log(\kappa)$ is within range both in its regression coefficient and $R^2$ value to what experimentalists consider as compliance with the power-law (see figure 1b). *Therefore this white Gaussian noise trajectory seems to fit the two-thirds power law model in equation (1) above*.

## 1.1 Problem formulation

For any regular planar curve parameterized with $t$, we get from the Frenet-Serret formulas (see [19])[1]:

$$\kappa(t) = \frac{|\ddot{x}\dot{y} - \dot{x}\ddot{y}|}{(\dot{x}^2 + \dot{y}^2)^{\frac{3}{2}}} = \frac{|\ddot{x}\dot{y} - \dot{x}\ddot{y}|}{v^3(t)} \tag{3}$$

where $v(t) = \sqrt{\dot{x}^2 + \dot{y}^2}$. Denoting $\alpha(t) = |\ddot{x}\dot{y} - \dot{x}\ddot{y}|$ we obtain:

$$v(t) = \alpha(t)^{\frac{1}{3}} \cdot \kappa(t)^{-\frac{1}{3}} \tag{4}$$

or, in log-space[2]:

$$\log(v(t)) = \frac{1}{3}\log(\alpha(t)) - \frac{1}{3}\log(\kappa(t)) \tag{5}$$

Given a trajectory for which $\alpha$ is constant, the power-law in equation (1) above is obtained exactly (the term $\alpha$ is in fact the affine velocity of [17],[18], and thus a trajectory that yields a constant $\alpha$ would mean movement at constant affine velocity).

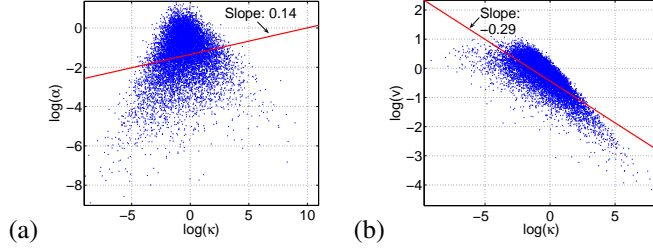

(a)    (b)

Figure 1: Given a trajectory composed of normally distributed position data with constant time intervals, we calculate and plot: **(a)** $\log(\alpha)$ versus $\log(\kappa)$ and **(b)** $\log(v)$ versus $\log(\kappa)$ with their linear regression lines. The correlation coefficient in (a) is 0.14, entailing the one in (b) to be -0.29 (see text). Moreover, the $R^2$ value in (a) is 0.04, much smaller than in the 0.57 value in (b).

Denoting the linear regression coefficient of $\log(v)$ versus $\log(\kappa)$ by $\beta$ and the linear regression coefficient of $\log(\alpha)$ versus $\log(\kappa)$ by $\xi$, it can be easily shown that (5) entails:

$$\beta = -\frac{1}{3} + \frac{\xi}{3} \qquad (6)$$

Hence, if $\log(\alpha)$ and $\log(\kappa)$ are statistically uncorrelated, the linear regression coefficient between them, which we termed $\xi$, would be 0, and thus from (6) the linear regression coefficient of $\log(v)$ versus $\log(\kappa)$, which we named $\beta$, would be exactly $-\frac{1}{3}$. Therefore, any trajectory that produces $\log(\alpha)$ and $\log(\kappa)$ that are statistically uncorrelated would precisely conform to the power-law in (1)[3].

If $\log(\alpha)$ and $\log(\kappa)$ are weakly correlated, such that $\xi$ (the linear regression coefficient of $\log(\alpha)$ versus $\log(\kappa)$) is small, the effect on $\beta$ (the linear regression coefficient of $\log(v)$ versus $\log(\kappa)$) would result in a positive offset of $\frac{\xi}{3}$ from the $-\frac{1}{3}$ value of the power-law. Below, we analyze $\xi$ for random position data, and show that it is indeed small and that $\beta$ takes values close to $-\frac{1}{3}$. Figure 1 portrays a typical $\log(v)$ versus $\log(\kappa)$ linear regression plot for the case of a trajectory composed of random data sampled from an i.i.d. normal distribution.

## 2  Power-law analysis for trajectories composed of normally distributed samples

Let us take the time-series $(x_i, y_i)_{i=1}^{n}$ where $x_i, y_i \sim N(0, 1)$, i.i.d. Let $t_i$ denote the time at sample $i$ and for all $i$ let $t_{i+1} - t_i = 1$. From this time series we calculate $\alpha$, $\kappa$ and $v$ by central finite differences[4]. Again, we denote the linear regression coefficient of $\log(\alpha)$ versus $\log(\kappa)$ by $\xi$, and thus $\widehat{\log(\alpha)} = \text{const} + \xi \log(\kappa)$, where $\xi = \frac{\text{Covariance}_{[\log(\kappa), \log(\alpha)]}}{\text{Variance}_{[\log(\kappa)]}}$. And from (6) we know that a linear regression of $\log(v)$ versus $\log(\kappa)$ would result in $\beta = -\frac{1}{3} + \frac{\xi}{3}$.

The fact that $\xi$ is scaled down three-fold to give the offset of $\beta$ from $-\frac{1}{3}$ is significant. It means that for $\beta$ to achieve values far from $-\frac{1}{3}$, $\xi$ would need to be very big. For example,

in order for $\beta$ to be 0 (i.e. motion at constant tangential speed), $\xi$ would need to be 1, which requires perfect correlation between $\log(\alpha)$, which is a time-dependant variable, and $\log(\kappa)$, which is a geometric one. This could be taken to suggest that for a control system to maintain movement at $\beta$ values that are remote from $-\frac{1}{3}$ would require some non-trivial control of the correlation between $\log(\alpha)$ and $\log(\kappa)$.

Running 100 Monte-Carlo simulations[5], each drawing a time series of 1,000,000 normally distributed points, we estimated $\xi = 0.1428 \pm 0.0013$ ($R^2 = 0.0357 \pm 0.0006$). $\xi$'s magnitude and its corresponding $R^2$ value suggest that $\log(\alpha)$ and $\log(\kappa)$ are only weakly correlated (hence the ball-like shape in Figure 1a). The same type of simulations gave $\beta = -0.2857 \pm 0.0004$ ($R^2 = 0.5715 \pm 0.0011$), as expected. Both $\beta$ and its $R^2$ magnitudes are within what is considered by experimentalists to be the range of applicable values for the power-law. Moreover, standard outlier detection and removal techniques as well as robust linear regression make $\beta$ approach closer to $-\frac{1}{3}$ and increase the $R^2$ value.

Measurements of human drawing movements in 3D also exhibit the power-law ([20],[16]). We therefore decided to repeat the same analysis procedure for 3D data (i.e. drawing time-series $(x_i, y_i, z_i)_{i=1}^{n}$ i.i.d. from $N(0,1)$, and extracting $v$, $\alpha$ and $\kappa$ according to their 3D definitions). This time we obtained $\xi = -0.0417 \pm 0.0009$ ($R^2 = 0.0036 \pm 0.0002$) and as expected $\beta = -0.3472 \pm 0.0003$ ($R^2 = 0.6944 \pm 0.0006$). This is even closer to the power-law values, as defined in (1).

This phenomenon also occurs when we repeat the procedure for trajectories composed of uniformly distributed samples with constant time intervals. The linear regression of $\log(v)$ versus $\log(\kappa)$ for planar trajectories gives $\beta = -0.2859 \pm 0.0004$ ($R^2 = 0.5724 \pm 0.0009$). 3D trajectories of uniformly distributed samples give us $\beta = -0.3475 \pm 0.0003$ ($R^2 = 0.6956 \pm 0.0007$) under the same simulation procedure. In both cases the parameters obtained for the uniform distribution are very close to those of the normal distribution.

## 3 Analysis of signal and noise combinations

### 3.1 Original (Non-filtered) signal and noise combinations

Another interesting question has to do with the combination of signal and noise. Every experimentally measured signal has some noise incorporated in it, be it measurement-device noise or noise internal to the human motor system. But how much noise must be present to transform a signal that does not conform to the power-law to one that does? We took a planar ellipse with a major axis of 0.35m and minor axis of 0.13m (well within the standard range of dimensions used as templates for measuring the power-law for humans, see figure 2b), and spread 120 equispaced samples over its perimeter (a typical number of samples for the sampling rate given below). The time intervals were constant at 0.01s (100 Hz is of the order of magnitude of contemporary measurement equipment). This elliptic trajectory is thus traversed at constant speed, despite not having constant curvature. It therefore does not obey the power-law (a "sanity check" of our simulations gave $\beta = -0.0003$, $R^2 = 0.0028$).

At this stage, normally distributed noise with various standard-deviations was added to this ellipse. We ran 100 simulations for every noise magnitude and averaged the power-law parameters $\beta$ and $R^2$ obtained from $\log(v)$ versus $\log(\kappa)$ linear regressions for each noise magnitude (see figure 2a). The level of noise required to drive the non-power-law-compliant trajectory to obey the power-law is rather small; a standard deviation of about

*http://www.cs.huji.ac.il/~urim/NIPS_2005/Monte_Carlo.m*

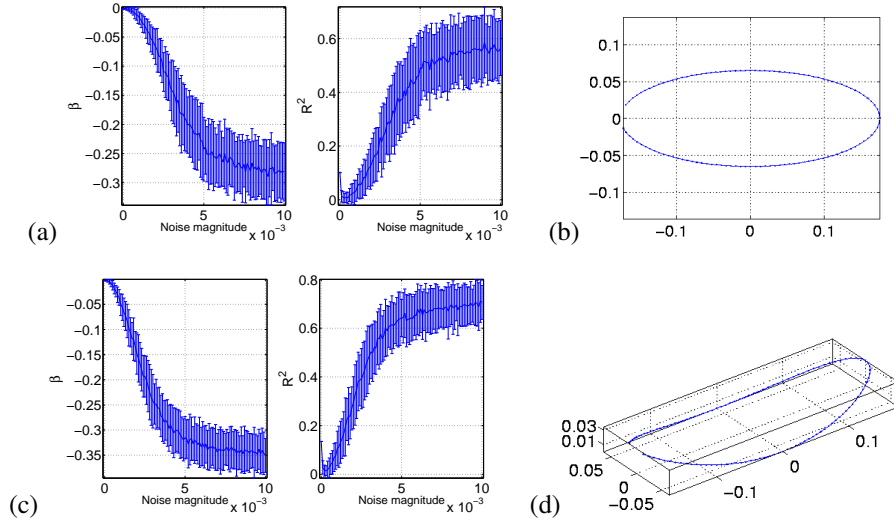

Figure 2: **(a)** $\beta$ and $R^2$ values of the power-law fit for trajectories composed of the non-power-law planar ellipse given in (b) combined with various magnitudes of noise (portrayed as the standard deviations of the normally distributed noise that was added). **(c)** $\beta$ and $R^2$ values for the non-power-law 3D bent ellipse given in (d). All distances are measured in meters.

0.005m is sufficient[6].

The same procedure was performed for a 3D bent-ellipse of similar proportions and perimeter in order to test the effects of noise on spatial trajectories (see figure 2c and d). We placed the samples on this bent ellipse in an equispaced manner, so that it would be traversed at constant speed (and indeed $\beta = -0.0042$, $R^2 = 0.1411$). This time the standard deviation of the noise which was required for power-law-like behavior in 3D was 0.003m or so, a bit smaller than in the planar case. Naturally, had we chosen smaller ellipses, less noise would have been required to make them obey the power-law (for instance, if we take a 0.1 by 0.05m ellipse, the same effect would be obtained with noise of about 0.002m standard deviation for the planar case and 0.0015m for a 3D bent ellipse of the same magnitude). Note that both for the planar and spatial shapes, the noise-level that drives the non-power-law signal to conform to the power-law is in the order of magnitude of the average displacement between consecutive samples.

### 3.2 Does filtering solve the problem?

All the analysis above was for raw data, whereas it is common practice to low-pass filter experimentally obtained trajectories before extracting $\kappa$ and $v$. If we take a non-power-law signal, contaminate it with enough noise for it to comply with the power-law and then filter it, would the resulting signal obey the power-law or not? We attempted to answer this question by contaminating the constant-speed bent-ellipse of the previous subsection (reminder: $\beta = -0.0003$, $R^2 = 0.0028$) with Gaussian noise of standard deviation 0.005m. This resulted in a trajectory with $\beta = -0.3154 \pm 0.0323$ ($R^2 = 0.6303 \pm 0.0751$) for 100 simulation runs (actually a bit closer to the power-law than the noise alone). We then low-pass filtered each trajectory with a zero-lag second-order Butterworth filter with 10

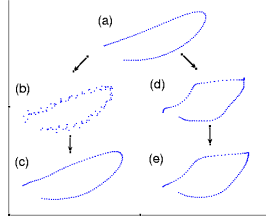

Figure 3: a graphical outline of Procedure 1 (from the top to the bottom left) and Procedure 2 (from the top to the bottom right). **(a)** The original non-power-law signal **(b)** Signal in (a) plus white noise **(c)** Signal in (b) after smoothing. **(d)** Signal in (a) with added correlated noise **(e)** Signal in (d) after smoothing. All signals but (a) and (c) obey the power-law.

Hz cutoff frequency. This returned a signal essentially without power-law compliance, i.e. $\beta = -0.0472 \pm 0.0209$ ($R^2 = 0.1190 \pm 0.0801$) . Let us name this process *Procedure 1*.

But what if the noise at hand is more resistant to smoothing? Taking 3D Gaussian noise and smoothing it (using the same type of Butterworth filtering as above) does not make the resulting signal any less compliant to the power-law. Monte-Carlo simulations of smoothed random trajectories (100 repetitions of 1,000,000 samples each) resulted in $\beta = -0.3473 \pm 0.0003$ ($R^2 = 0.6945 \pm 0.0007$), which is the same as the original noise (which had $\beta = -0.3472 \pm 0.0003$, $R^2 = 0.6944 \pm 0.0006$)[7]. We therefore ran the signal plus noise simulations again, this time adding smoothed-noise (increasing its magnitude five-fold to compensate for the loss of energy of this noise due to the filtering) to the constant-speed bent-ellipse. This time the combined signal yielded a power-law fit of $\beta = -0.3175 \pm 0.0414$ ($R^2 = 0.6260 \pm 0.0798$), leaving it power-law compliant. However, this time the same filtering procedure as above left us with a signal that could still be considered to obey the power-law, with $\beta = -0.2747 \pm 0.0481$ ($R^2 = 0.5498 \pm 0.0698$). We name this process *Procedure 2*. Procedures 1 and 2 are portrayed graphically in figure 3. If we continue and increase the noise magnitude the effect of the smoothing at the end of Procedure 2 becomes less apparent, with the smoothed trajectories sometimes conforming to the power-law (mainly in terms of $R^2$) better than before the smoothing.

### 3.3 Levels of noise inherent to upper limb movement in human data

We conducted a preliminary experiment to explore the level of noise intrinsic to the human motor system. Subjects were instructed to repetitively and continuously trace ellipses in 3D while seated with their trunk restrained to the back of a rigid chair and their eyes closed (to avoid spatial cues in the room, see [16]). They were to suppose that there exists a spatial elliptical pattern before them, which they were to traverse with their hand. The 3D hand position was recorded at 100 Hz using NDI's Optotrak 2010. Given their goal, it is reasonable to assume that the variance between the trajectories in the different iterations is composed of measurement noise as well as noise internal to the subject's motor systems (since the inter-repetition drift was removed by PCA alignment after segmenting the continuous motion into underlying iterations). We thus analyzed the recorded time series in order to find that variance (see figure 4a) and compared this to the average variance of the synthetic equispaced bent ellipse combined with correlated noise (see figure 4b). While more careful experiments may be needed to extract the exact SNR of human limb movements, it appears that the level of noise in human limb movement is comparable to the level of noise that can cause power-law behavior even for non power-law signals.

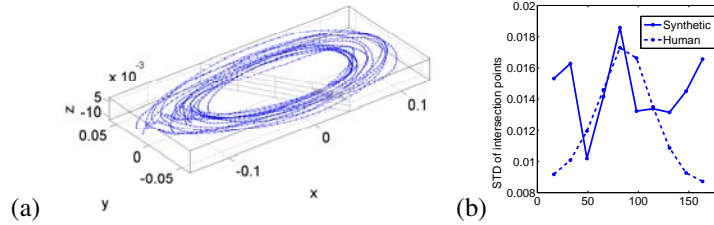

(a)     (b)

Figure 4: Noise level in repetitive upper limb movement. The variance of the different iterations was measured at 10 different positions along the ellipse, defined by a plane passing through the origin, perpendicular to the first two principle components of the ellipses. The angles between every two neighboring planes were equal. **(a)** For each position, the intersection of each iteration of the trajectory with the plane was calculated. **(b)** The standard deviation of the different intersections of each plane was measured, and is depicted for synthetic and human data.

## 4   Discussion

We do not suggest that the power-law, which stems from analysis of human data, is a bogus phenomenon, resulting only from measurement noise. Yet our results do suggest caution when carrying out experiments that either aim to verify the power-law or assume its existence. When performing such experimentation, one should always be sure to verify that the signal-to-noise ratio in the system is well within the bounds where it does not drive the results toward the power-law. It might further be wise to conduct Monte-Carlo simulations with the specific parameters of the problem to ascertain this. If we focus on the measurement device noise alone, it should be noted that whereas many modern devices for planar motion measurement tend to have a measurement accuracy superior to the 0.002m or so (which we have shown to be enough to produce power-law from noise), the same cannot be said for contemporary 3D measurement devices. There, errors of magnitudes in the order of about 0.002m can certainly occur. In addition, one should keep in mind that even for smaller noise magnitudes some drift toward the power-law does occur. This must be taken into consideration when analyzing the results. Last, muscle-tremor must also be borne in mind as another source of noise, especially when dealing with pathologies.

Moreover, following the results above, it is clear that when a significant amount of noise is incorporated into the system, simply applying an off-the-shelf smoothing procedure would not necessarily satisfactorily remove it, especially if it is correlated (i.e. not white). More-over, the smoothing procedure will most likely distort the signal to some degree, even if the noise is white. Therefore smoothing is not an easy "magic cure" for the power-law-from-noise phenomenon.

Another interesting aspect of our results has to do with the light they shed on the origins of the power-law. Previous works showed that the power-law can be derived from smoothness criteria for human trajectories, be it the assumption that these minimize the end-point's jerk ([14]), jerk along a predefined path ([15]) or variability due to noise inherent in the motor system itself ([12]), or that the power law is due to smoothing inherent in the human motor system (especially the muscles, [21]) or to smooth joint oscillations ([16]). The results presented here suggest the opposite might be true as well. The power-law can be derived from the noise itself, which is inherent in our motor system (and which is likely to be correlated noise), rather than from any smoothing mechanisms which damp it.

**Acknowledgements**

This research was supported in part by the HFSPO grant to T.F.; E.P. is supported by an Eshkol

fellowship of the Israeli Ministry of Science.

## Footnotes

[1]Though there exists a definition for signed curvature for planar curves (i.e. without the absolute value in the numerator of equation 3), we refer to the absolute value of the curvature, as done in the power-law. Therefore, in our case, $\kappa(t)$ is the absolute value of the instantaneous curvature.

[2]Non-linear regression in (4) should naturally be performed instead of log-space linear regression in (5). However, this linear regression in log-space is the method of choice in the motor-control literature, despite the criticism of [16]. We therefore opted for it here as well.

[3]$\alpha$ being constant is naturally a special case of uncorrelated $\log(\alpha)$ and $\log(\kappa)$.

[4]We used the central finite differencing technique here mainly for ease of use and analysis. Other differentiation techniques, either utilizing more samples (e.g. the Lagrange 5-point method) or analytic differentiation of smoothing functions (e.g. smoothing splines), yielded similar results. In a more general sense, smoothing techniques introduce local correlations (between neighboring samples in time) into the trajectory. Yet globally, for large time series, the correlation remains weak.

[5]The Matlab code for this simple simulation can be found at:

[6]0.005m is about half the distance between consecutive samples in this trajectory and sampling rate.

[7]This result goes hand in hand with what was said before. Introducing local correlations between samples does not alter the power-law-from-noise phenomenon.

# References

[1] F. Lacquaniti, C. Terzuolo, and P. Viviani. The law relating kinematic and figural aspects of drawing movements. *Acta Psychologica*, 54:115–130, 1983.

[2] P. Viviani. Do units of motor action really exist. *Experimental Brain Research*, 15:201–216, 1986.

[3] P. Viviani and M. Cenzato. Segmentation and coupling in complex movements. *Journal of Experimental Psychology: Human Perception and Performance*, 11(6):828–845, 1985.

[4] P. Viviani and R. Schneider. A developmental study of the relationship between geometry and kinematics in drawing movements. *Journal of Experimental Psychology*, 17:198–218, 1991.

[5] J. T. Massey, J. T. Lurito, G. Pellizzer, and A. P. Georgopoulos. Three-dimensional drawings in isometric conditions: relation between geometry and kinematics. *Experimental Brain Research*, 88(3):685–690, 1992.

[6] A. B. Schwartz. Direct cortical representation of drawing. *Science*, 265(5171):540–542, 1994.

[7] C. deSperati and P. Viviani. The relationship between curvature and velocity in two dimensional smooth pursuit eye movement. *The Journal of Neuroscience*, 17(10):3932–3945, 1997.

[8] P. Viviani and N. Stucchi. Biological movements look uniform: evidence of motor perceptual interactions. *Journal of Experimental Psychology: Human Perception and Performance*, 18(3):603–626, 1992.

[9] P. Viviani, G. Baud Bovoy, and M. Redolfi. Perceiving and tracking kinesthetic stimuli: further evidence of motor perceptual interactions. *Journal of Experimental Psychology: Human Perception and Performance*, 23(4):1232–1252, 1997.

[10] S. Kandel, J. Orliaguet, and P. Viviani. Perceptual anticipation in handwriting: The role of implicit motor competence. *Perception and Psychophysics*, 62(4):706–716, 2000.

[11] S. Vieilledent, Y. Kerlirzin, S. Dalbera, and A. Berthoz. Relationship between velocity and curvature of a human locomotor trajectory. *Neuroscience Letters*, 305(1):65–69, 2001.

[12] C. M. Harris and D. M. Wolpert. Signal-dependent noise determines motor planning. *Nature*, 394(6695):780–784, 1998.

[13] P. Viviani and T. Flash. Minimum-jerk, two-thirds power law, and isochrony: converging approaches to movement planning. *Journal of Experimental Psychology: Human Perception and Performance*, 21(1):32–53, 1995.

[14] M. J. Richardson and T. Flash. Comparing smooth arm movements with the two-thirds power law and the related segmented-control hypothesis. *Journal of Neuroscience*, 22(18):8201–8211, 2002.

[15] E. Todorov and M. Jordan. Smoothness maximization along a predefined path accurately predicts the speed profiles of complex arm movements. *Journal of Neurophysiology*, 80(2):696–714, 1998.

[16] S. Schaal and D. Sternad. Origins and violations of the 2/3 power law in rhythmic three-dimensional arm movements. *Experimental Brain Research*, 136(1):60–72, 2001.

[17] A. A. Handzel and T. Flash. Geometric methods in the study of human motor control. *Cognitive Studies*, 6:1–13, 1999.

[18] F. E. Pollick and G. Sapiro. Constant affine velocity predicts the 1/3 power law of planar motion perception and generation. *Vision Research*, 37(3):347–353, 1997.

[19] J. Oprea. Differential geometry and its applications. *Prentice-Hall*, 1997.

[20] U. Maoz and T. Flash. Power-laws of three-dimensional movement. *Unpublished manuscript*.

[21] P. L. Gribble and D. J. Ostry. Origins of the power law relation between movement velocity and curvature: modeling the effects of muscle mechanics and limb dynamics. *Journal of Neurophysiology*, 76:2853–2860, 1996.
